# From Stochastic Nonlinear Integrate-and-Fire to Generalized Linear Models

**Skander Mensi**
School of Computer and Communication Sciences and Brain-Mind Institute
Ecole Polytechnique Federale de Lausanne
1015 Lausanne EPFL, SWITZERLAND
skander.mensi@epfl.ch

**Richard Naud**
School of Computer and Communication Sciences and Brain-Mind Institute
Ecole Polytechnique Federale de Lausanne
1015 Lausanne EPFL, SWITZERLAND
richard.naud@epfl.ch

**Wulfram Gersnter**
School of Computer and Communication Sciences and Brain-Mind Institute
Ecole Polytechnique Federale de Lausanne
1015 Lausanne EPFL, SWITZERLAND
wulfram.gerstner@epfl.ch

## Abstract

Variability in single neuron models is typically implemented either by a stochastic Leaky-Integrate-and-Fire model or by a model of the Generalized Linear Model (GLM) family. We use analytical and numerical methods to relate state-of-the-art models from both schools of thought. First we find the analytical expressions relating the subthreshold voltage from the Adaptive Exponential Integrate-and-Fire model (AdEx) to the Spike-Response Model with escape noise (SRM as an example of a GLM). Then we calculate numerically the link-function that provides the firing probability given a deterministic membrane potential. We find a mathematical expression for this link-function and test the ability of the GLM to predict the firing probability of a neuron receiving complex stimulation. Comparing the prediction performance of various link-functions, we find that a GLM with an exponential link-function provides an excellent approximation to the Adaptive Exponential Integrate-and-Fire with colored-noise input. These results help to understand the relationship between the different approaches to stochastic neuron models.

## 1   Motivation

When it comes to modeling the intrinsic variability in simple neuron models, we can distinguish two traditional approaches. One approach is inspired by the stochastic Leaky Integrate-and-Fire (LIF) hypothesis of Stein (1967) [1], where a noise term is added to the system of differential equations implementing the leaky integration to a threshold. There are multiple versions of such a stochastic LIF [2]. How the noise affects the firing probability is also a function of the parameters of the neuron model. Therefore, it is important to take into account the refinements of simple neuron models in terms of subthreshold resonance [3, 4], spike-triggered adaptation [5, 6] and non-linear spike

initiation [7, 5]. All these improvements are encompassed by the Adaptive Exponential Integrate-and-Fire model (AdEx [8, 9]).

The other approach is to start with some deterministic dynamics for the the state of the neuron (for instance the instantaneous distance from the membrane potential to the threshold) and link the probability intensity of emitting a spike with a non-linear function of the state variable. Under some conditions, this type of model is part of a greater class of statistical models called Generalized Linear Models (GLM [10]). As a single neuron model, the Spike Response Model (SRM) with escape noise is a GLM in which the state variable is explicitly the distance between a deterministic voltage and the threshold. The original SRM could account for subthreshold resonance, refractory effects and spike-frequency adaptation [11]. Mathematically similar models were developed independently in the study of the visual system [12] where spike-frequency adaptation has also been modeled [13]. Recently, this approach has retained increased attention since the probabilistic framework can be linked with the Bayesian theory of neural systems [14] and because Bayesian inference can be applied to the population of neurons [15].

In this paper, we investigate the similarity and differences between the state-of-the-art GLM and the stochastic AdEx. The motivation behind this work is to relate the traditional threshold neuron models to Bayesian theory. Our results extend the work of Plesser and Gerstner (2000) [16] since we include the non-linearity for spike initiation and spike-frequency adaptation. We also provide relationships between the parameters of the AdEx and the equivalent GLM. These precise relationships can be used to relate analog implementations of threshold models [17] to the probabilistic models used in the Bayesian approach.

The paper is organized as follows: We first describe the expressions relating the SRM state-variable to the parameters of the AdEx (Sect. 3.1) in the subthreshold regime. Then, we use numerical methods to find the non-linear link-function that models the firing probability (Sect. 3.2). We find a functional form for the SRM link-function that best describes the firing probability of a stochastic AdEx. We then compare the performance of this link-function with the often used exponential or linear-rectifier link-functions (also called half-wave linear rectifier) in terms of predicting the firing probability of an AdEx under complex stimulus (Sect. 3.3). We find that the exponential link-function yields almost perfect prediction. Finally, we explore the relations between the statistic of the noise and the sharpness of the non-linearity for spike initiation with the parameters of the SRM.

## 2 Presentation of the Models

In this section we present the general formula for the stochastic AdEx model (Sect. 2.1) and the SRM (Sect 2.2).

### 2.1 The Stochastic Adaptive Exponential Integrate-and-Fire Model

The voltage dynamics of the stochastic AdEx is given by:

$$\tau_m \dot{V} = E_l - V + \Delta_T \exp\left(\frac{V - \Theta}{\Delta_T}\right) - Rw + RI + R\epsilon \tag{1}$$

$$\tau_w \dot{w} = a(V - E_l) - w \tag{2}$$

where $\tau_m$ is the membrane time constant, $E_l$ the reverse potential, $R$ the membrane resistance, $\Theta$ is the threshold, $\Delta_T$ is the shape factor and $I(t)$ the input current which is chosen to be an Ornstein-Uhlenbeck process with correlation time-constant of 5 ms. The exponential term $\Delta_T \exp(\frac{V - \Theta}{\Delta_T})$ is a non-linear function responsible for the emission of spikes and $\epsilon$ is a diffusive white noise with standard deviation $\sigma$ (i.e. $\epsilon \sim \mathcal{N}(0, \sigma)$). Note that the diffusive white-noise does not imply white noise fluctuations of the voltage $V(t)$, the probability distribution of $V(t)$ will depend on $\Delta_T$ and $\Theta$. The second variable, $w$, describes the subthreshold as well as the spike-triggered adaptation both parametrized by the coupling strength $a$ and the time constant $\tau_w$. Each time $\hat{t}_j$ the voltage goes to infinity, we assumed that a spike is emitted. Then the voltage is reset to a fixed value $V_r$ and $w$ is increased by a constant value $b$.

### 2.2 The Generalized Linear Model

In the SRM, The voltage $V(t)$ is given by the convolution of the injected current $I(t)$ with the membrane filter $\kappa(t)$ plus the additional kernel $\eta(t)$ that acts after each spikes (here we split the

spike-triggered kernel in two $\eta(t) = \eta_v(t) + \eta_w(t)$ for reasons that will become clear later):

$$V(t) \quad = \quad E_l + [\kappa * I](t) + \sum_{\{\hat{t}_j\}} \left( \eta_v(t - \hat{t}_j) + \eta_w(t - \hat{t}_j) \right) \tag{3}$$

Then at each time $\hat{t}_j$ a spike is emitted which results in a change of voltage described by $\eta(t) = \eta_v(t) + \eta_w(t)$.

Given the deterministic voltage, (Eq. 3) a spike is emitted according to the firing intensity $\lambda(V)$:

$$\lambda(t) \quad = \quad f(V(t)) \tag{4}$$

where $f(\cdot)$ is an arbitrary function called the *link-function*. Then the firing behavior of the SRM depends on the choice of the link-function and its parameters. The most common link-function used to model single neuron activities are the linear-rectifier and the exponential function.

## 3    Mapping

In order to map the stochastic AdEx to the SRM we follow a two-step procedure. First we derive the filter $\kappa(t)$ and the kernels $\eta_v(t)$ and $\eta_w(t)$ analytically as a function of AdEx parameters. Second, we derive the link-function of the SRM from the stochastic spike emission of the AdEx.

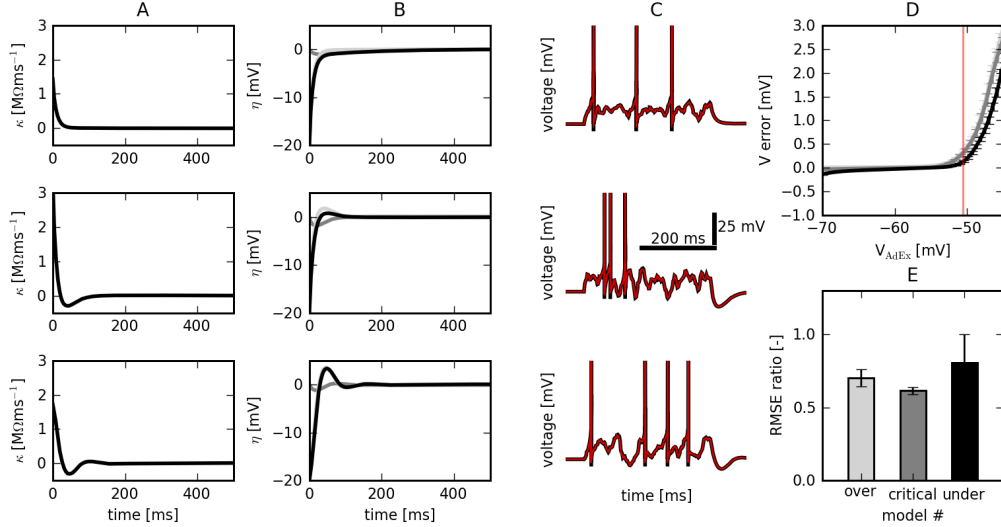

Figure 1: Mapping of the subthreshold dynamics of an AdEx to an equivalent SRM. A. Membrane filter $\kappa(t)$ for three different sets of parameters of the AdEx leading to over-damped, critically damped and under-damped cases (upper, middle and lower panel, respectively). B. Spike-Triggered $\eta(t)$ (black), $\eta_v(t)$ (light gray) and $\eta_w$ (gray) for the three cases. C. Example of voltage trace produced when an AdEx is stimulated with a step of colored noise (black). The corresponding voltage from a SRM stimulated with the same current and where we forced the spikes to match those of the AdEx (red). D. Error in the subthreshold voltage ($V_{AdEx} - V_{GLM}$) as a function of the mean voltage of the AdEx, for the three different cases: over-, critically and under-damped (light gray, gray and black, respectively) with $\Delta_T = 1$ mV. Red line represents the voltage threshold $\Theta$. E. Root Mean Square Error (RMSE) ratio for the three cases with $\Delta_T = 1$ mV. The RMSE ratio is the RMSE between the deterministic $V_{SRM}$ and the stochastic $V_{AdEx}$ divided by the RMSE between repetitions of the stochastic AdEx voltage. The error bar shows a single standard deviation as the RMSE ratio is averaged accross multiple value of $\sigma$.

### 3.1    Subthreshold voltage dynamics

We start by assuming that the non-linearity for spike initiation does not affect the mean subthreshold voltage of the stochastic AdEx (see Figure 1 D). This assumption is motivated by the small $\Delta_T$

observed in *in-vitro* recordings (from 0.5 to 2 mV [8, 9]) which suggest that the subthreshold dynamics are mainly linear except very close to $\Theta$. Also, we expect that the non-linear link-function will capture some of the dynamics due to the non-linearity for spike initiation. Thus it is possible to rewrite the deterministic subthreshold part of the AdEx (Eq. 1-2 without $\epsilon$ and without $\Delta_T \exp((V - \Theta)/\Delta_T)$) using matrices:

$$\dot{\mathbf{x}} = A\mathbf{x} \tag{5}$$

$$\text{with } \mathbf{x} = \begin{pmatrix} V \\ w \end{pmatrix} \text{ and } A = \begin{bmatrix} -\frac{1}{\tau_m} & -\frac{1}{g_l \tau_m} \\ \frac{a}{\tau_w} & -\frac{1}{\tau_w} \end{bmatrix} \tag{6}$$

In this form, the dynamics of the deterministic AdEx voltage is a damped oscillator with a driving force. Depending on the eigenvalues of $A$ the system could be over-damped, critically damped or under-damped. The filter $\kappa(t)$ of the GLM is given by the impulse response of the system of coupled differential equations of the AdEx, described by Eq. 5 and 6. In other words, one has to derive the response of the system when stimulating with a Dirac-delta function. The type of damping gives three different qualitative shapes of the kernel $\kappa(t)$, which are summarized in Table 3.1 and Figure 1 A. Since the three different filters also affect the nature of the stochastic voltage fluctuations, we will keep the distinction between over-damped, critically damped and under-damped scenarios throughout the paper. This means that our approach is valid for at least 3 types of diffusive voltage-noise (i.e. the white noise $\epsilon$ in Eq. 1 filtered by 3 different membrane filters $\kappa(t)$).

To complete the description of the deterministic voltage, we need an expression for the spike-triggered kernels. The voltage reset at each spike brings a spike-triggered jump in voltage of magnitude $\Delta = V_r - V(\hat{t})$. This perturbation is superposed to the current fluctuations due to $I(t)$ and can be mediated by a Delta-diract pulse of current. Thus we can write the voltage reset kernel by:

$$\eta_v(t) = \frac{\Delta}{\kappa(0)} [\delta * \kappa](t) = \frac{\Delta}{\kappa(0)} \kappa(t) \tag{7}$$

where $\delta(t)$ is the Dirac-delta function. The shape of this kernel depends on $\kappa(t)$ and can be computed from Table 3.1 (see Figure 1 B).

Finally, the AdEx mediates spike-frequency adaptation by the jump of the second variables $w$. From Eq. 2 we can see that this produces a current $w_{\text{spike}}(t) = b \exp(-t/\tau_w)$ that can cumulate over subsequent spikes. The effect of this current on voltage is then given by the convolution of $w_{\text{spike}}(t)$ with the membrane filter $\kappa(t)$. Thus in the SRM framework the spike-frequency adaptation is taken into account by:

$$\eta_w(t) \quad = \quad [w_{\text{spike}} * \kappa](t) \tag{8}$$

Again the precise form of $\eta_w(t)$ depends on $\kappa(t)$ and can be computed from Table 3.1 (see Figure 1 B).

At this point, we would like to verify our assumption that the non-linearity for spike emission can be neglected. Fig. 1 C and D shows that the error between the voltage from Eq. 3 and the voltage from the stochastic AdEx is generally small. Moreover, we see that the main contribution to the voltage prediction error is due to the mismatch close to the spikes. However the non-linearity for spike initiation may change the probability distribution of the voltage fluctuations, which in turn influences the probability of spiking. This will influence the choice of the link-function, as we will see in the next section.

## 3.2 Spike Generation

Using $\kappa(t)$, $\eta_v(t)$ and $\eta_w(t)$, we must relate the spiking probability of the stochastic AdEx as a function of its deterministic voltage. According to [2] the probability of spiking in time bin $dt$ given the deterministic voltage $V(t)$ is given by:

$$p(V) = \text{prob}\{\text{spike in } [\text{t}, \text{t} + \text{dt}]\} \quad = \quad 1 - \exp(-f(V(t))dt) \tag{9}$$

where $f(\cdot)$ gives the firing intensity as a function of the deterministic $V(t)$ (Eq. 3). Thus to extract the link-function $f$ we have to compute the probability of spiking given $V(t)$ for our SRM. To do so we apply the method proposed by Jolivet *et al.* (2004) [18], where the probability of spiking is simply given by the distribution of the deterministic voltage estimated at the spike times divided by the distribution of the SRM voltage when there is no spike (see figure 2 A). One can numerically compute these two quantities for our models using N repetitions of the same stimulus.

Table 1: Analytical expressions for the membrane filter $\kappa(t)$ in terms of the parameters of the AdEx for over-, critically-, and under-damped cases.

| Membrane Filter: $\kappa(t)$ | | |
| --- | --- | --- |
| over-damped if: | critically-damped if: | under-damped if: |
| $(\tau_m + \tau_w)^2 > \frac{4\tau_m\tau_w(g_l+a)}{g_l}$ | $(\tau_m + \tau_w)^2 = \frac{4\tau_m\tau_w(g_l+a)}{g_l}$ | $(\tau_m + \tau_w)^2 < \frac{4\tau_m\tau_w(g_l+a)}{g_l}$ |
| $\kappa(t) = k_1 e^{\lambda_1 t} + k_2 e^{\lambda_2 t}$ | $\kappa(t) = (\alpha t + \beta)e^{\lambda t}$ | $\kappa(t) = (k_1 \cos(\omega t) + k_2 \sin(\omega t))e^{\lambda t}$ |
| $\lambda_1 = \frac{1}{2\tau_m\tau_w}\left(-(\tau_m + \tau_w) + \sqrt{(\tau_m + \tau_w)^2 - 4\frac{\tau_m\tau_w}{g_l}(g_l + a)}\right)$ | $\lambda = \frac{-(\tau_m+\tau_w)}{2\tau_m\tau_w}$ | $\lambda = \frac{-(\tau_m+\tau_w)}{2\tau_m\tau_w}$ |
| $\lambda_2 = \frac{1}{2\tau_m\tau_w}\left(-(\tau_m + \tau_w) - \sqrt{(\tau_m + \tau_w)^2 - 4\frac{\tau_m\tau_w}{g_l}(g_l + a)}\right)$ | $\alpha = \frac{\tau_m-\tau_w}{2C\tau_m\tau_w}$ | $\omega = \sqrt{\left|\left(\frac{\tau_w-\tau_m}{2\tau_m\tau_w}\right)^2 - \frac{a}{g_l\tau_m\tau_w}\right|}$ |
| $k_1 = \frac{-(1+(\tau_m\lambda_2))}{C\tau_m(\lambda_1-\lambda_2)}$ | $\beta = \frac{1}{C}$ | $k_1 = \frac{1}{C}$ |
| $k_2 = \frac{1+(\tau_m\lambda_1)}{C\tau_m(\lambda_1-\lambda_2)}$ | | $k_2 = \frac{-(1+\tau_m\lambda)}{C\omega\tau_m}$ |

The standard deviation $\sigma$ of the noise and the parameter $\Delta_T$ of the AdEx non-linearity may affect the shape of the link-function. We thus extract $p(V)$ for different $\sigma$ and $\Delta_T$ (Fig. 2 B). Then using visual heuristics and previous knowledge about the potential analytical expression of the link-funtion, we try to find a simple analytical function that captures $p(V)$ for a large range of combinations of $\sigma$ and $\Delta_T$. We observed that the $\log(-\log(p))$ is close to linear in most studied conditions Fig. 2 B suggesting the following two distributions of $p(V)$:

$$p(V) = 1 - \exp\left(-\exp\left(\frac{V - V_T}{\Delta V}\right)\right) \tag{10}$$

$$p(V) = \exp\left(-\exp\left(-\frac{V - V_T}{\Delta V}\right)\right) \tag{11}$$

Once we have $p(V)$, we can use Eq. 4 to obtain the equivalent SRM link-function, which leads to:

$$f(V) = \frac{-1}{dt}\log\left(1 - p(V)\right) \tag{12}$$

Then the two potential link-functions of the SRM can be derived from Eq. 10 and Eq. 11 (respectively):

$$f(V) = \lambda_0 \exp\left(\frac{V - V_T}{\Delta V}\right) \tag{13}$$

$$f(V) = -\lambda_0 \log\left(1 - \exp\left(-\exp\left(-\frac{V - V_T}{\Delta V}\right)\right)\right) \tag{14}$$

with $\lambda_0 = \frac{1}{dt}$, $V_T$ the threshold of the SRM and $\Delta V$ the sharpness of the link-function (i.e. the parameters that governs the degree of the stochasticity). Note that the exact value of $\lambda_0$ has no importance since it is redundant with $V_T$. Eq. 13 is the standard exponential link-function, but we call Eq. 14 the log-exp-exp link-function.

### 3.3 Prediction

The next point is to evaluate the fit quality of each link-function. To do this, we first estimate the parameters $V_T$ and $\Delta V$ of the GLM link-function that maximize the likelihood of observing a spike

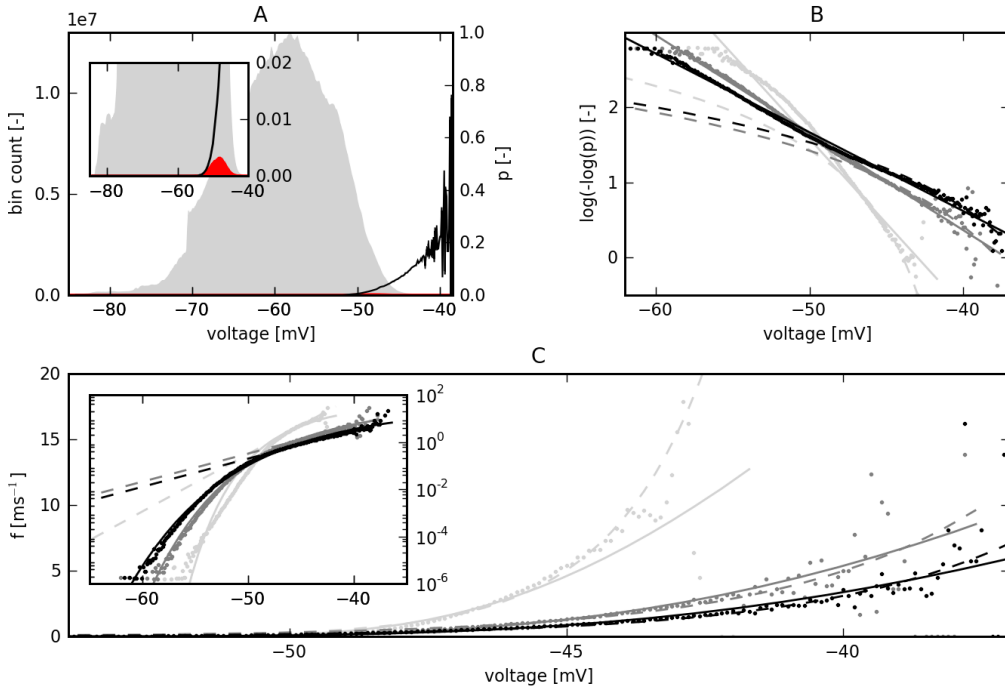

Figure 2: SRM link-function. A. Histogram of the SRM voltage at the AdEx firing times (red) and at non-firing times (gray). The ratio of the two distributions gives $p(V)$ (Eq. 9, dashed lines). Inset, zoom to see the voltage histogram evaluated at the firing time (red). B. $\log(-\log(p))$ as a function of the SRM voltage for three different noise levels $\sigma = 0.07, 0.14, 0.18$ nA (pale gray, gray, black dots, respectively) and $\Delta_T = 1$ mV. The line is a linear fit corresponding to the log-exp-exp link-function and the dashed line corresponds to a fit with the exponential link-function. C. Same data and labeling scheme as B, but plotting $f(V)$ according to Eq. 12. The lines are produced with Eq. 14 with parameters fitted as described in B. and the dashed lines are produced with Eq. 13. Inset, same plot but on a semi-log(y) axis.

train generated with an AdEx. Second we look at the predictive power of the resulting SRM in terms of Peri-Stimulus Time Histogram (PSTH). In other words we ask how close the spike trains generated with a GLM are from the spike train generated with a stochastic AdEx when both models are stimulated with the same input current.

For any GLM with link-function $f(V) \equiv f(t|I,\theta)$ and parameters $\theta$ regulating the shape of $\kappa(t)$, $\eta_v(t)$ and $\eta_w(t)$, the Negative Log-Likelihood (NLL) of observing a spike-train $\{\hat{t}\}$ is given by:

$$\text{NLL} \quad = \quad -\left(\sum_{\hat{t}} \log(f(t|I,\theta)) - \sum_t f(t|I,\theta)\right) \tag{15}$$

It has been shown that the negative log-likelihood is convex in the parameters if $f$ is convex and log-concave [19]. It is easy to show that a linear-rectifier link-function, the exponential link-function and the log-exp-exp link-function all satisfy these conditions. This allows efficient estimation of the optimal parameters $\hat{V}_T$ and $\hat{\Delta V}$ using a simple gradient descent. One can thus estimate from a single AdEx spike train the optimal parameters of a given link-function, which is more efficient than the method used in Sect. 3.2.

The minimal NLL resulting from the gradient descent gives an estimation of the fit quality. A better estimate of the fit quality is given by the distance between the PSTHs in response to stimuli not used for parameter fitting . Let $\nu_1(t)$ be the PSTH of the AdEx, and $\nu_2(t)$ be the PSTH of the fitted SRM,

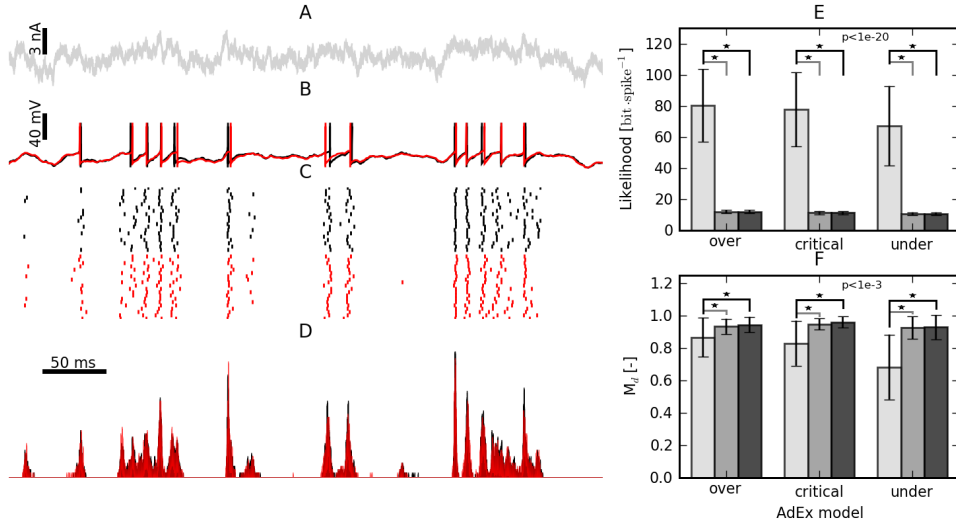

Figure 3: PSTH prediction. A. Injected current. B. Voltage traces produced by an AdEx (black) and the equivalent SRM (red), when stimulated with the current in A. C. Raster plot for 20 realizations of AdEx (black tick marks) and equivalent SRM (red tick marks). D. PSTH of the AdEx (black) and the SRM (red) obtained by averaging 10,000 repetitions. E. Optimal log-likelihood for the three cases of the AdEx, using three different link-functions, a linear-rectifier (light gray), an exponential link-function (gray) and the link-function defined by Eq. 14 (dark gray), these values are obtained by averaging over 40 different combinations $\sigma$ and $\Delta_T$ (see Fig. 4). Error bars are one standard deviation, the stars denote a significant difference, two-sample t-test with $\alpha = 0.01$. F. same as E. but for $M_d$ (Eq. 16).

then we use $M_d \in [0, 1]$ as a measure of match:

$$M_d = \frac{2 \int \left(\nu_1(t) - \nu_2(t)\right)^2 dt}{\int \nu_1(t)^2 dt + \int \nu_2(t)^2 dt} \tag{16}$$

$M_d = 1$ means that it is impossible to differentiate the SRM from the AdEx in terms of their PSTHs, whereas a $M_d$ of 0 means that the two PSTHs are completely different. Thus $M_d$ is a normalized similarity measure between two PSTHs. In practice, $M_d$ is estimated from the smoothed (boxcar average of 1 ms half-width) averaged spike train of 1 000 repetitions for each models. We use both the NLL and $M_d$ to quantify the fit quality for each of the three damping cases and each of the three link-functions.

Figure 3 shows the match between the stochastic AdEx used as a reference and the derived GLM when both are stimulated with the same input current (Fig. 3 A). The resulting voltage traces are almost identical (Fig. 3 B) and both models predict almost the same spike trains and so the same PSTHs (Fig. 3 C and D). More quantitalively, we see on Fig. 3 E and F, that the linear-rectifier fits significantly worse than both the exponential and log-exp-exp link-functions, both in terms of NLL and of $M_d$. The exponential link-function performs as well as the log-exp-exp link-function, with a spike train similarity measure $M_d$ being almost 1 for both.

Finally the likelihood-based method described above gives us the opportunity to look at the relationship between the AdEx parameters $\sigma$ and $\Delta_T$ that governs its spike emission and the parameters $V_T$ and $\Delta V$ of the link-function (Fig. 4). We observe that an increase of the noise level produces a flatter link-function (greater $\Delta V$) while an increase in $\Delta_T$ also produces an increase in $\Delta V$ and $V_T$ (note that Fig. 4 shows $\Delta V$ and $V_T$ for the exponential link-function only, but equivalent results are obtained with the log-exp-exp link-function).

## 4   Discussion

In Sect. 3.3 we have shown that it is possible to predict with almost perfect accuracy the PSTH of a stochastic AdEx model using an appropriate set of parameters in the SRM. Moreover, since

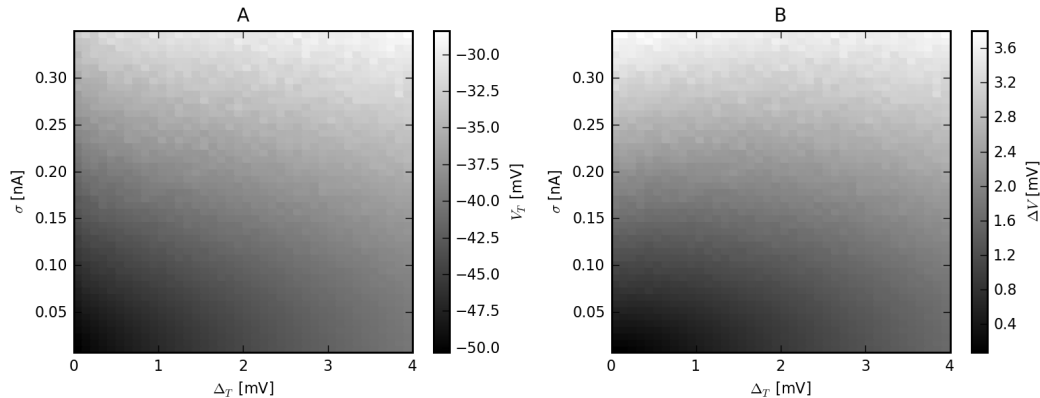

Figure 4: Influence of the AdEx parameters on the parameters of the exponential link-function. A. $V_T$ as a function of $\Delta_T$ and $\sigma$. B. $\Delta V$ as a function of $\Delta_T$ and $\sigma$.

the subthreshold voltage of the AdEx also gives a good match with the deterministic voltage of the SRM, we expect that the AdEx and the SRM will not differ in higher moments of the spike train probability distributions beyond the PSTH. We therefore conclude that diffusive noise models of the type of Eq. 1-2 are equivalent to GLM of the type of Eq. 3-4. Once combined with similar results on other types of stochastic LIF (*e.g.* correlated noise), we could bridge the gap between the literature on GLM and the literature on diffusive noise models.

Another noteworthy observation pertains to the nature of the link-function. The link-function has been hypothesized to be a linear-rectifier, an exponential, a sigmoidal or a Gaussian [16]. We have observed that for the AdEx the link-function follows Eq. 14 that we called the log-exp-exp link-function. Although the link-function is log-exp-exp for most of the AdEx parameters, the exponential link-function gives an equivalently good prediction of the PSTH. This can be explained by the fact that the difference between log-exp-exp and exponential link-functions happens mainly at low voltage (i.e. far from the threshold), where the probability of emitting a spike is so low (Figure 2 C, until -50 mv). Therefore, even if the exponential link-function overestimates the firing probability at these low voltages it rarely produces extra spikes. At voltages closer to the threshold, where most of the spikes are emitted, the two link-functions behave almost identically and hence produce the same PSTH. The Gaussian link-function can be seen as lying in-between the exponential link-function and the log-exp-exp link-function in Fig. 2. This means that the work of Plesser and Gerstner (2000) [16] is in agreement with the results presented here. The importance of the time-derivative of the voltage stressed by Plesser and Gerstner (leading to a two-dimensional link-function $f(V, \dot{V})$) was not studied here to remain consistent with the typical usage of GLM in neural systems [14].

Finally we restricted our study to exponential non-linearity for spike initiation and do not consider other cases such as the Quadratic Integrate-and-fire (QIF, [5]) or other polynomial functional shapes. We overlooked these cases for two reasons. First, there are many evidences that the non-linearity in neurons (estimated from *in-vitro* recordings of Pyramidal neurons) is well approximated by a single exponential [9]. Second, the exponential non-linearity of the AdEx only affects the subthreshold voltage at high voltage (close to threshold) and thus can be neglected to derive the filters $\kappa(t)$ and $\eta(t)$. Polynomial non-linearities on the other hand affect a larger range of the subthreshold voltage so that it would be difficult to justify the linearization of subthreshold dynamics essential to the method presented here.

# References

[1] R. B. Stein, "Some models of neuronal variability," *Biophys J*, vol. 7, no. 1, pp. 37–68, 1967.

[2] W. Gerstner and W. Kistler, *Spiking neuron models.* Cambridge University Press New York, 2002.

[3] E. Izhikevich, "Resonate-and-fire neurons," *Neural Networks*, vol. 14, no. 883-894, 2001.

[4] M. J. E. Richardson, N. Brunel, and V. Hakim, "From subthreshold to firing-rate resonance," *Journal of Neurophysiology*, vol. 89, pp. 2538–2554, 2003.

[5] E. Izhikevich, "Simple model of spiking neurons," *IEEE Transactions on Neural Networks*, vol. 14, pp. 1569–1572, 2003.

[6] S. Mensi, R. Naud, M. Avermann, C. C. H. Petersen, and W. Gerstner, "Parameter extraction and classification of three neuron types reveals two different adaptation mechanisms," Under review.

[7] N. Fourcaud-Trocme, D. Hansel, C. V. Vreeswijk, and N. Brunel, "How spike generation mechanisms determine the neuronal response to fluctuating inputs," *Journal of Neuroscience*, vol. 23, no. 37, pp. 11 628–11 640, 2003.

[8] R. Brette and W. Gerstner, "Adaptive exponential integrate-and-fire model as an effective description of neuronal activity," *Journal of Neurophysiology*, vol. 94, pp. 3637–3642, 2005.

[9] L. Badel, W. Gerstner, and M. Richardson, "Dependence of the spike-triggered average voltage on membrane response properties," *Neurocomputing*, vol. 69, pp. 1062–1065, 2007.

[10] P. McCullagh and J. A. Nelder, *Generalized linear models*, 2nd ed. Chapman & Hall/CRC, 1998, vol. 37.

[11] W. Gerstner, J. van Hemmen, and J. Cowan, "What matters in neuronal locking?" *Neural computation*, vol. 8, pp. 1653–1676, 1996.

[12] D. Hubel and T. Wiesel, "Receptive fields and functional architecture of monkey striate cortex," *Journal of Physiology*, vol. 195, pp. 215–243, 1968.

[13] J. Pillow, L. Paninski, V. Uzzell, E. Simoncelli, and E. Chichilnisky, "Prediction and decoding of retinal ganglion cell responses with a probabilistic spiking model," *Journal of Neuroscience*, vol. 25, no. 47, pp. 11 003–11 013, 2005.

[14] K. Doya, S. Ishii, A. Pouget, and R. P. N. Rao, *Bayesian brain: Probabilistic approaches to neural coding*. The MIT Press, 2007.

[15] S. Gerwinn, J. H. Macke, M. Seeger, and M. Bethge, "Bayesian inference for spiking neuron models with a sparsity prior," in *Advances in Neural Information Processing Systems*, 2007.

[16] H. Plesser and W. Gerstner, "Noise in integrate-and-fire neurons: From stochastic input to escape rates," *Neural Computation*, vol. 12, pp. 367–384, 2000.

[17] J. Schemmel, J. Fieres, and K. Meier, "Wafer-scale integration of analog neural networks," in *Neural Networks, 2008. IJCNN 2008. (IEEE World Congress on Computational Intelligence). IEEE International Joint Conference on*, june 2008, pp. 431 –438.

[18] R. Jolivet, T. Lewis, and W. Gerstner, "Generalized integrate-and-fire models of neuronal activity approximate spike trains of a detailed model to a high degree of accuracy," *Journal of Neurophysiology*, vol. 92, pp. 959–976, 2004.

[19] L. Paninski, "Maximum likelihood estimation of cascade point-process neural encoding models," *Network: Computation in Neural Systems*, vol. 15, pp. 243–262, 2004.

